# Logic and MRF Circuitry for Labeling Occluding and Thinline Visual Contours

**Eric Saund**
Palo Alto Research Center
3333 Coyote Hill Rd.
Palo Alto, CA 94304
`saund@parc.com`

## Abstract

This paper presents representation and logic for labeling contrast edges and ridges in visual scenes in terms of both surface occlusion (border ownership) and thinline objects. In natural scenes, thinline objects include sticks and wires, while in human graphical communication thinlines include connectors, dividers, and other abstract devices. Our analysis is directed at both natural and graphical domains. The basic problem is to formulate the logic of the interactions among local image events, specifically contrast edges, ridges, junctions, and alignment relations, such as to encode the natural constraints among these events in visual scenes. In a sparse heterogeneous Markov Random Field framework, we define a set of interpretation nodes and energy/potential functions among them. The minimum energy configuration found by Loopy Belief Propagation is shown to correspond to preferred human interpretation across a wide range of prototypical examples including important illusory contour figures such as the Kanizsa Triangle, as well as more difficult examples. In practical terms, the approach delivers correct interpretations of inherently ambiguous hand-drawn box-and-connector diagrams at low computational cost.

## 1   Introduction

A great deal of attention has been paid to the curious phenomenon of illusory contours in visual scenes [5]. The most famous example is the Kanizsa Triangle (Figure 1). Although a number of explanations have been proposed, computational accounts have converged on the understanding that illusory contours are an outcome of the more general problem of labeling scene contours in terms of causal events such as surface overlap. Illusory contours are the visual system's way of expressing belief in an occlusion relation between two surfaces having the same lightness and therefore lacking a visible contrast edge. The phenomena are interesting in their revelation of interactions among multiple factors comprising the visual system's prior assumptions about what constitutes likely interpretations of ambiguous input.

Several computational models for this process have generated interpretations of Kanizsa-like figures corresponding to human perception. Williams[9] formulated an integer-linear

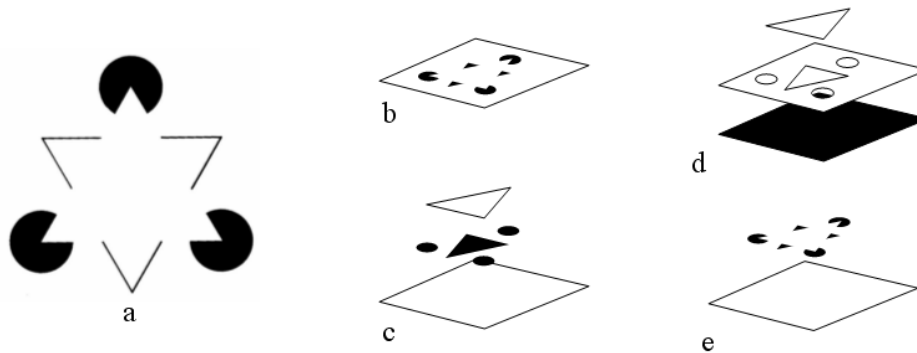

Figure 1: a. Original Kanizsa Triangle. b. Solid surface version. c. Human preferred interpretation. d, e. Other valid interpretations.

optimization problem with hard constrains originating from the topology of contours and junctions, and soft constraints representing figural biases for non-accidental interpretations and figural closure. Heitger and von der Heydt[2] implemented a series of nonlinear filtering operations that enacted interactions among line terminations and junctions to infer modal completions corresponding to illusory contours. Geiger[1] used a dense Markov Random Field to represent surface depths explicitly and propagated local evidence through a diffusion process. Saund[6] enumerated possible generic and non-generic interpretations of T- and L-junctions to set up an optimization problem solved by deterministic annealing. Liu and Wang[4] set up a network of contours traversing the boundaries of segmented regions, which interact to propagate local information through an iterative updating scheme.

This paper expands this body of previous work in the following ways:

- The computational model is expressed in terms of a sparse heterogeneous Markov Random Field whose solution is accessible to fast techniques such as Loopy Belief Propagation.
- We introduce interpretations of thinlines in addition to solid surfaces, adding a significant layer of richness and complexity.
- The model infers occlusion relations of surfaces depicted by line drawings of their borders, as well as solid graphics depictions.
- We devise MRF energy functions that implement circuitry for sophisticated logical constraints of the domain.

The result is a formulation that is both fast and effective at correctly interpreting a greater range of psychophysical and near-practical contour configuration examples than has heretofor been demonstrated. The model exposes aspects of fundamental ambiguity to be resolved by the incorporation of additional constraints and domain-specific knowledge.

## 2 Interpretation Nodes and Relations

### 2.1 Visible Contours and Contour Ends

Early vision studies commonly distinguish several models for visible contour creation and measurement, including contrast edges, lines or ridges, ramps, color and texture edges, etc. Let us idealize to consider only contrast edges and ridges (also known as "bars"), measured at a single scale. We include in our domain of interest human-generated graphical

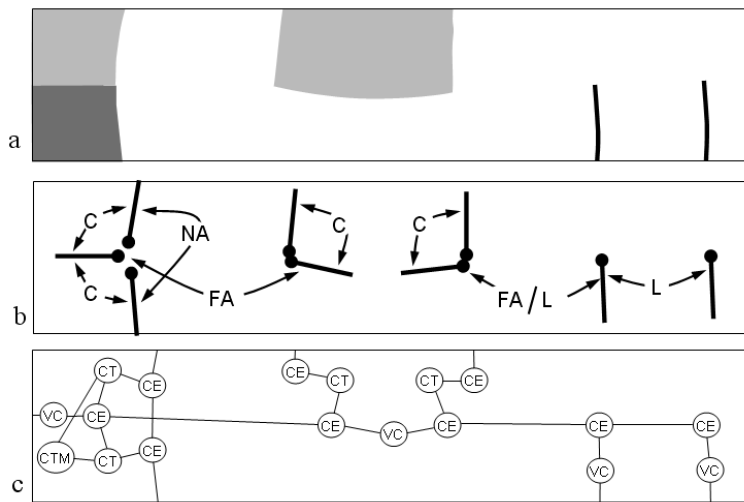

Figure 2: a. Sample image region. b. Spatial relation categories characterizing links in the MRF among Contour End nodes: Corner, Near Alignment, Far Alignment, Lateral. c. Resulting MRF including nodes of type Visible Contour, Contour End, Corner Tie, and Corner Tie Mediator.

figures. Contrast edges arise from distinct regions or surfaces, while ridges may represent either a boundary between regions or else a "thinline", i.e. a physical or graphical object whose shape is essentially defined by a one-dimensional path at our scale of measurement. Examples of thinlines in photographic imagery include twigs, sidewalk cracks, and telephone wires, while in graphical images thinlines include separators, connectors, and arrow shafts. Figure 7e shows a hand-drawn sketch in which some lines (measured as ridges) are intended to define boxes and therefore represent region boundaries, while others are connectors between boxes. We take the contour interpretation problem to include the analysis of this type of scene in addition to classical illusory contour figures.

For any input data, we may construct a Markov Random Field consisting of four types of nodes derived from measured contrast edge and ridge contours. An interpretation is an assignment of states to nodes. Local potentials and the potential matrices associated with pairwise links between nodes encode constraints and biases among interpretation states based on the spatial relations among the visible contours. Figure 2 illustrates MRF nodes types and links for a simple example input image, as explained below.

Let us assume that contours defining region boundaries are assigned an occlusion direction, equivalent to relative surface depth and hence boundary ownership. Figure 3 shows the possible mappings between visible image contours measured as contrast edges or ridges, and their interpretation in terms of direction of surface overlap or else thinline object. Contrast edges always correspond to surface occlusion, while ridges may represent either a surface boundary or a thinline object. Correspondingly, the simplest MRF node type is the Visible Contour node which has state dimension 3 corresponding to two possible overlap directions and one thinline interpretation.

Most of the interesting evidence and interaction occurs at terminations and junctions of visible contours. Contour End nodes are given the job of explaining why a smooth visible edge or ridge contour has terminated visibility, and hence they will encode the bulk of the modal (illusory) and amodal (occluded) completion information of a computed interpretation. Smooth visible contours may terminate in four ways:

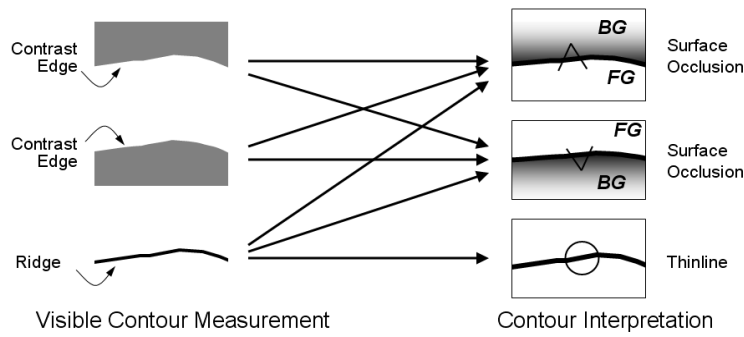

Figure 3: Permissible mappings between visible edge and ridge contours and interpretations. Wedges indicate direction of surface overlap: white (FG) surface occludes shaded (BG) surface.

1. The surface boundary contour or thinline object changes direction (turns a corner)

2. The contour becomes modal because the background surface lacks a visible edge with the foreground surface.

3. The contour becomes amodal because it becomes occluded by another surface.

4. The contour simply terminates when an surface overlap meets the end of a fold, or when a thin object or graphic stops.

Contour Ends therefore have 3x4 = 12 interpretation states as shown in Figure 4.

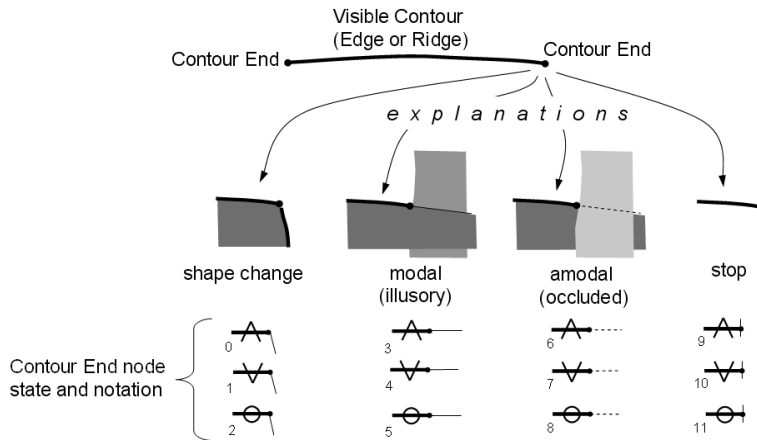

Figure 4: Contour End nodes have state dimension 12 indicating contour overlap type/direction (overlap or thinline) and one of four explanations for termination of the visible contour.

Every Visible Contour node is linked to its two corresponding Contour End nodes through energy matrices (or equivalently, potential matrices, using Potential $\psi = \exp^{-E}$) representing simple compatibility among overlap direction/thinline interpretation states. Additional links in the network are created based on spatial relations among Contour Ends as described next.

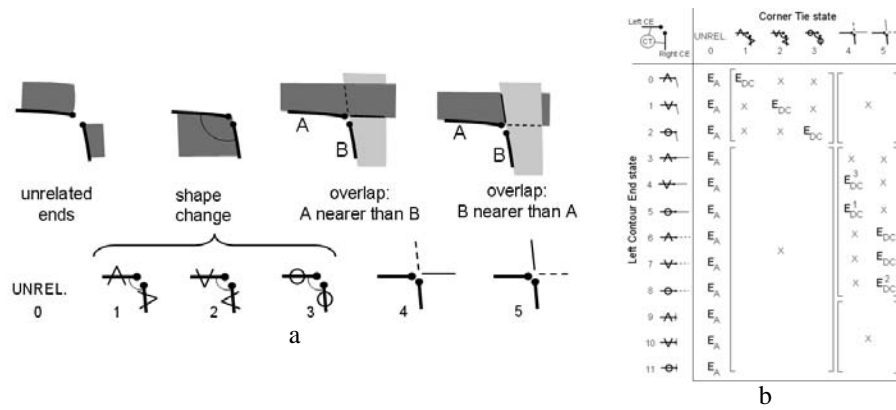

Figure 5: a. Corner Tie nodes have state dimension 6 indicating the causal relationship between the Contour End nodes they link. b. Energy matrix linking the Left Contour End of a pair of corner-relation Contour Ends to their Corner Tie. X indicates high energy prohibiting the state combination. $E_A$ refers to a low penalty for Accidental Coincidence of the Contour Ends. $E_{DC}$ refers to a (typically low) penalty of two Contour Ends failing to meet the ideal geometrical constraints of meeting at a corner. The subscripts refer to necessary Near-Alignment Relations on the Contour Ends. The energy matrix linking the Right End Contour to the Corner Tie swaps the 5th and 6th columns.

## 2.2 Contour Ends Relation Links

Let us consider five classes of pairwise geometric relations among observed contour ends: Corner, Near-Alignment, Far-Alignment, Lateral, and Unrelated. Mathematical expressions forming the bases for these relations may be engineered as measures of distance and smooth continuation such as used by Saund [6]. The Corner relation depends only on proximity; Near-Alignment depends on proximity and alignment; Far-Alignment omits the proximity requirement.

Within this framework a further refinement distinguishes ridge Contour Ends from those arising from contrast edges. Namely, ridge ends are permitted to form Lateral relation links which correspond to potential modal contours. Contrast edge Contour Ends are excluded from this link type because they terminate at junctions which distribute modal and amodal completion roles to their participating Contour Ends. Contour End nodes from ridge contours may participate in Far-Alignment links but their local energies are set to preclude them from taking states representing modal completions.

In this way the present model fixes the topology of related ends in the process of setting up the Markov Graph. An important problem for future research is to formulate the Markov Graph to include *all* plausible Contour End pairings and have the actual pairings sort themselves out at solution time.

Biases about preferred and less-preferred interpretations are represented through the terms in the energy matrices linking related Contour Ends. In accordance with prior work, we bias energy terms associated with curved Visible Contours and junctions of Contour Ends in favor of convex object interpretations. Space limitations preclude presenting the energy matrices in detail, but we discuss the main novel and significant considerations.

The simplest case is pairs of Contour Ends sharing a Near-Alignment or Far-Alignment relation. These energy matrices are constructed to trade off priors regarding accidental alignment versus amodal or modal invisible contour completion interpretations. For Con-

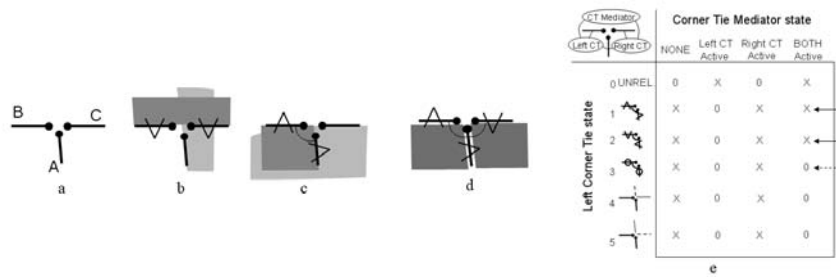

Figure 6: The Corner Tie Mediator node restricts border ownership of occluding contours to physically consistent interpretations. The energy matrix shown in e links the Corner Tie Mediator to the Left Corner Tie of a pair sharing a Contour End. X indicates high energy. The energy matrix for the link to the Right Corner Tie swaps the second and third columns.

tour End pairs that are relatively near and well aligned, energy terms corresponding to causally unrelated interpretations (CE states 0,1,2) are large, while terms corresponding to amodal completion with compatible overlap/thinline property (CE states 6,7,8) are small. Actual energy values for the matrices are assigned by straightforward formulas derived from the Proximity and Smooth Continuation terms mentioned above. Per Kanizsa, modal completion interpretations (CE states 3,4,5) are somewhat more expensive than amodal interpretations, by a constant factor. Energy terms shift their relative weights in favor of causally unrelated interpretations (CE corner states 0,1,2) as the Contour Ends become more distant and less aligned.

Contour Ends sharing a Corner relation can be related in one of three ways: they can be causally unrelated and unordered in depth; they can represent a turning of a surface boundary or thinline object; they can represent overlap of one contour above the other. In order to exploit the geometry of Contour Ends as local evidence, these alternatives must be articulated and entered into the MRF node graph. To do this we therefore introduce a third type of node, the Corner Tie node, possessing six states as illustrated in Figure 5a.

The energy matrix relating Contour End nodes and Corner Tie nodes is shown in Figure 5b. It contains low energy terms representing the Corner Tie's belief that the Contour End termination is due to direction change (turning a corner). It also contains low energy terms representing the conditions of one Contour End's owning surface overlapping the other contour, i.e. the relative depth relation between these contours in the scene.

### 2.3 Constraints on Overlaps and Thinlines at Junctions

Physical considerations impose hard constraints on the interpretations of End Pairs meeting at a junction. Consider the T-junction in Figure 6a. One preferred interpretation for a T-junction is occlusion (6b). A less-preferred but possible interpretation is a change of direction (corner) by one surface, with accidental alignment by another contour (6c). What is *impossible* is for a surface boundary to bifurcate and "belong" to both sides of the T (6d).

This type of constraint cannot be enforced by the purely pairwise Corner Tie node. We therefore introduce a fourth node type, the Corner Tie Mediator. This node governs the number of Corner Ties that any Contour End can claim to form a direction change (corner turn) relation with. The energy matrix for the Corner Tie Mediator node is shown in Figure 6e: multiple Corner-Ties in the overlap direction-turn states (CT states 1 & 2) are excluded (solid arrows). But note that the matrix contains a low energy term (dashed arrow) for the formation of multiple direction-turn Corner-Ties provided they are in the Thinline state (CT state 3); branching of thinline objects *is* physically permissible.

# 3   Experiments and Conclusion

Loopy Belief Propagation under the Max-Product algorithm seeks the MAP configuration which is equivalent to the minimum-energy assignment of states [8]. We have not encountered a failure of LBP to converge, and it is quite rare to encounter a lower-energy assignment of states than the algorithm delivers starting from an initial uniform distribution over states. However, multiple stable fixed points can exist. For some ambiguous figures such as Figure 7e in which qualitatively different interpretations have similar energies, one may clamp one or more nodes to alternative states, leading to LBP solutions which persist once the clamping is removed. This invites the exploration of N-best configuration solution techniques [10].

Figure 7 demonstrates MAP assignments corresponding to preferred human interpretations of the classic Kanizsa illusory contour figure and others containing both aligning L-junction and ridge termination evidence for modal contours, amodal completions, and thinline objects. Note that the MRF correctly predicts that outline drawings of surface boundaries do not induce illusory contours.

Figure 7g borrows from experiments by Szummer and Cowans[7] toward a practical application in line drawing interpretation, in which closed boxes define regions while connectors remain interpreted as thinline objects. For this scene containing 369 nodes and 417 links, the entire process of forming the MRF and performing 100 iterations of LBP takes less than a second. The major pressures operating in these situations are a figural bias toward interpreting closed paths as convex regions, and a preference to interpret ridge contours participating in T- and X- junctions as thinline objects.

We have shown how explicit consideration of ridge features and thinline interpretations brings new complexity to the logic of sorting out depth relations in visual scenes. This investigation suggests that a sparse heterogeneous Markov Random Field approach may provide a suitable basis for such models.

## References

[1] Geiger, D., Kumaran, K, & Parida, L. (1996) Visual organization for figure/ground separation. in *Proc. IEEE CVPR* pp. 155-160.

[2] Heitger, F., & von der Heydt, R. (1993) A Computational Model of Neural Contour Processing: Figure-Ground Segregation and Illusory Contours. *Proc. ICCV '93*.

[3] Kanizsa, G. (1979) *Organization in Vision,* Praeger, New York.

[4] Liu, X., Wang, D. (2000) Perceptual Organization Based on Temporal Dynamics. in S.A. Solla, T.K. Leen, K.-R. Muller (eds.), *Advances in Neural Information Processing Systems 12*, pp. 38-44. MIT Press.

[5] Petry, S., & Meyer, G. (eds.) (1987) *The Perception of Illusory Contours,* Springer-Verlag, New York.

[6] Saund, E. (1999) Perceptual Organization of Occluding Contours of Opaque Surfaces, *CVIU* V. 76, No. 1, pp. 70-82.

[7] Szummer, M., & Cowans, P. (2004) Incorporating Context and User Feedback in Pen-Based Interfaces. *AAAI TR FS-04-06* (Papers from the 2004 AAAI Fall Symposium.)

[8] Weiss, Y., and Freeman, W.T. (2001) On the optimality of solutions of the max-product belief propagation algorithm in arbitrary graphs, *IEEE Trans. Inf. Theory* 47:2, pp. 723-735.

[9] Williams, L. (1990) Perceptual Organization of Occluding Contours. *Proc. ICCV '90.* pp. 639-649.

[10] Yanover, C. and Weiss, Y. (2003) Finding the M Most Probable Configurations Using Loopy Belief Propagation. in S. Thrun, L. Saul and B. Schölkpf, eds., *Advances in Neural Information Processing Systems 16*, MIT Press.

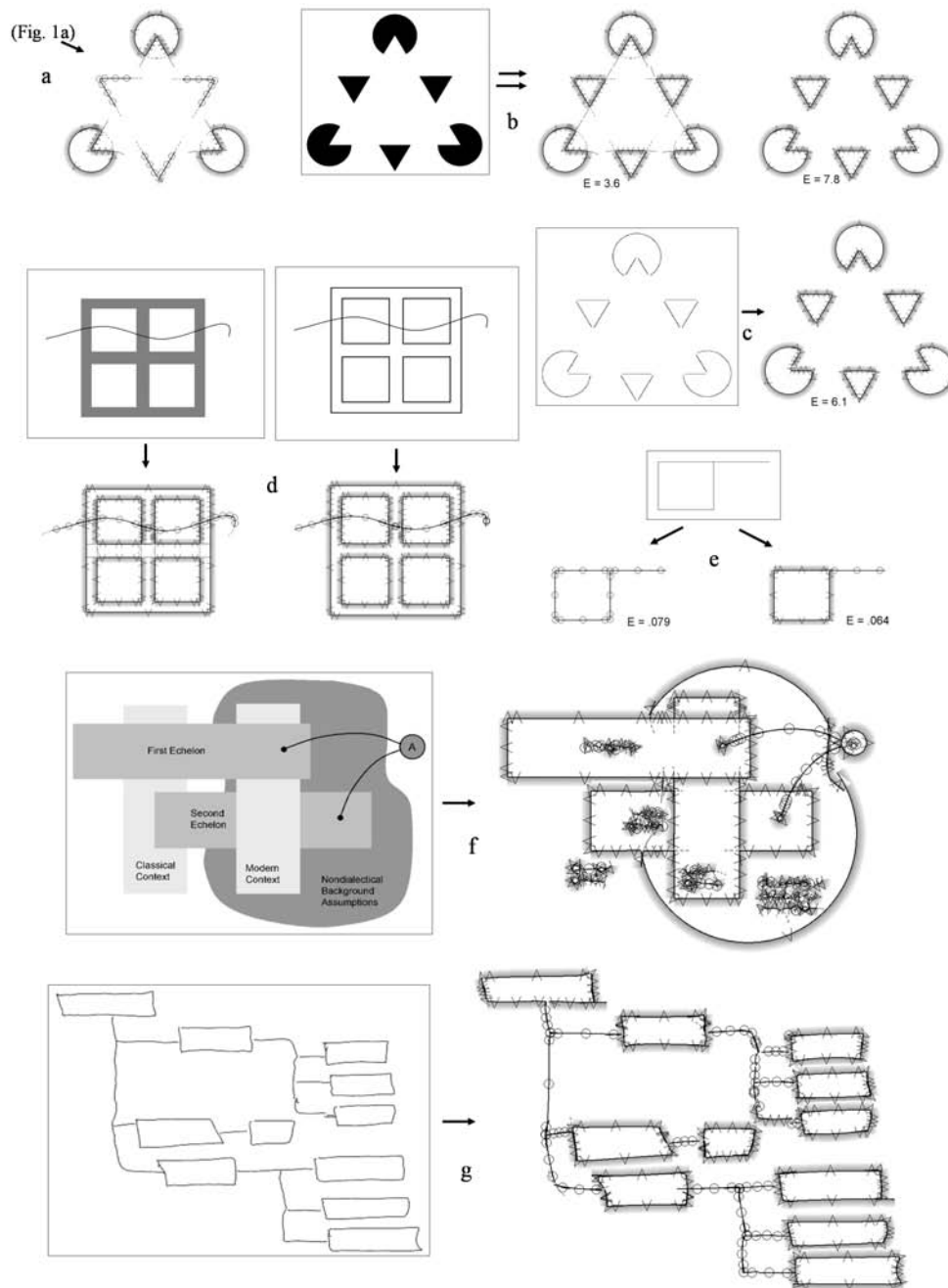

Figure 7: Experimental results. Arrow leads from input image to MRF interpretations. Curved contours are approximated by circular arcs. a. Original Kanizsa Triangle. b. Energies for preferred and "pac-man" interpretations. c. No illusory contour interpretations arise from outline drawing. d. Similar interpretations of solid and outline graphical windows overlain by a line. e. Similar energies obtain for an inherently ambiguous thinline/region figure. f. Graphical figure. g. Hand-drawn figure from [7] interpreted as closed regions and connectors.